# Efficient Inference for Distributions on Permutations

**Jonathan Huang**
Carnegie Mellon University
jch1@cs.cmu.edu

**Carlos Guestrin**
Carnegie Mellon University
guestrin@cs.cmu.edu

**Leonidas Guibas**
Stanford University
guibas@cs.stanford.edu

## Abstract

Permutations are ubiquitous in many real world problems, such as voting, rankings and data association. Representing uncertainty over permutations is challenging, since there are $n!$ possibilities, and typical compact representations such as graphical models cannot efficiently capture the mutual exclusivity constraints associated with permutations. In this paper, we use the "low-frequency" terms of a Fourier decomposition to represent such distributions compactly. We present *Kronecker conditioning*, a general and efficient approach for maintaining these distributions directly in the Fourier domain. Low order Fourier-based approximations can lead to functions that do not correspond to valid distributions. To address this problem, we present an efficient quadratic program defined directly in the Fourier domain to project the approximation onto a relaxed form of the marginal polytope. We demonstrate the effectiveness of our approach on a real camera-based multi-people tracking setting.

## 1 Introduction

Permutations arise naturally in a variety of real situations such as card games, data association problems, ranking analysis, etc. As an example, consider a sensor network that tracks the positions of $n$ people, but can only gather identity information when they walk near certain sensors. Such mixed-modality sensor networks are an attractive alternative to exclusively using sensors which can measure identity because they are potentially cheaper, easier to deploy, and less intrusive. See [1] for a real deployment. A typical tracking system maintains tracks of $n$ people and the identity of the person corresponding to each track. What makes the problem difficult is that identities can be confused when tracks cross in what we call mixing events. Maintaining accurate track-to-identity assignments in the face of these ambiguities based on identity measurements is known as the *Identity Management Problem* [2], and is known to be $NP$-hard. Permutations pose a challenge for probabilistic inference, because distributions on the group of permutations on $n$ elements require storing at least $n! - 1$ numbers, which quickly becomes infeasible as $n$ increases. Furthermore, typical compact representations, such as graphical models, cannot capture the mutual exclusivity constraints associated with permutations.

Diaconis [3] proposes maintaining a small subset of Fourier coefficients of the actual distribution allowing for a principled tradeoff between accuracy and complexity. Schumitsch et al. [4] use similar ideas to maintain a particular subset of Fourier coefficients of the log probability distribution. Kondor et al. [5] allow for general sets of coefficients, but assume a restrictive form of the observation model in order to exploit an efficient FFT factorization. The main contributions of this paper are:

- A new, simple and general algorithm, *Kronecker Conditioning*, which performs all probabilistic inference operations completely in the Fourier domain. Our approach is general, in the sense that it can address any transition model or likelihood function that can be represented in the Fourier domain, such as those used in previous work, and can represent the probability distribution with any desired set of Fourier coefficients.
- We show that approximate conditioning can sometimes yield Fourier coefficients which do not correspond to any valid distribution, and present a method for projecting the result back onto a relaxation of the marginal polytope.
- We demonstrate the effectiveness of our approach on a real camera-based multi-people tracking setting.

## 2 Filtering over permutations

In identity management, a permutation $\sigma$ represents a joint assignment of identities to internal tracks, with $\sigma(i)$ being the track belonging to the $i$th identity. When people walk too closely together, their identities can be confused, leading to uncertainty over $\sigma$. To model this uncertainty, we use a *Hidden Markov Model* on permutations, which is a joint distribution over $P(\sigma^{(1)}, \ldots, \sigma^{(T)}, z^{(1)}, \ldots, z^{(T)})$ which factors as:

$$P(\sigma^{(1)}, \ldots, \sigma^{(T)}, z^{(1)}, \ldots, z^{(T)}) = P(z^{(1)}|\sigma^{(1)}) \prod_t P(z^t|\sigma^{(t)}) \cdot P(\sigma^{(t)}|\sigma^{(t-1)}),$$

where the $\sigma^{(t)}$ are latent permutations and the $z^{(t)}$ denote observed variables. The conditional probability distribution $P(\sigma^{(t)}|\sigma^{(t-1)})$ is called the *transition model*, and might reflect for example, that the identities belonging to two tracks were swapped with some probability. The distribution $P(z^{(t)}|\sigma^{(t)})$ is called the *observation model*, which might capture a distribution over the color of clothing for each individual.

We focus on *filtering*, in which one queries the HMM for the posterior at some timestep, conditioned on all past observations. Given the distribution $P(\sigma^{(t)}|z^{(1)}, \ldots, z^{(t)})$, we recursively compute $P(\sigma^{(t+1)}|z^{(1)}, \ldots, z^{(t+1)})$ in two steps: a *prediction/rollup* step and a *conditioning* step. The first updates the distribution by multiplying by the transition model and marginalizing out the previous timestep: $P(\sigma^{(t+1)}|z^{(1)}, \ldots, z^{(t)}) = \sum_{\sigma^{(t)}} P(\sigma^{(t+1)}|\sigma^{(t)})P(\sigma^{(t)}|z^{(1)}, \ldots, z^{(t)})$. The second conditions the distribution on an observation $z^{(t+1)}$ using Bayes rule: $P(\sigma^{(t+1)}|z^{(1)}, \ldots, z^{(t+1)}) \propto P(z^{(t+1)}|\sigma^{(t+1)})P(\sigma^{(t+1)}|z^{(1)}, \ldots, z^{(t)})$. Since there are $n!$ permutations, a single update requires $O((n!)^2)$ flops and is consequently intractable for all but very small $n$. The approach that we advocate is to maintain a compact approximation to the true distribution based on the Fourier transform. As we discuss later, the Fourier based approximation is equivalent to maintaining a set of low-order marginals, rather than the full joint, which we regard as being analogous to an *Assumed Density Filter* [6].

## 3 Fourier projections of functions on the Symmetric Group

Over the last 50 years, the Fourier Transform has been ubiquitously applied to everything digital, particularly with the invention of the Fast Fourier Transform. On the real line, the Fourier Transform is a well-studied method for decomposing a function into a sum of sine and cosine terms over a spectrum of frequencies. Perhaps less familiar, is its group theoretic generalization, which we review in this section with an eye towards approximating functions on the group of permutations, the *Symmetric Group*. For permutations on $n$ objects, the Symmetric Group will be abbreviated by $S_n$. The formal definition of the Fourier Transform relies on the theory of group representations, which we briefly discuss first. Our goal in this section is to motivate the idea that the Fourier transform of a distribution $P$ is related to certain marginals of $P$. For references on this subject, see [3].

**Definition 1.** A *representation* of a group $G$ is a map $\rho$ from $G$ to a set of invertible $d_\rho \times d_\rho$ matrix operators which preserves algebraic structure in the sense that for all $\sigma_1, \sigma_2 \in G$, $\rho(\sigma_1 \sigma_2) = \rho(\sigma_1) \cdot \rho(\sigma_2)$. The matrices which lie in the image of this map are called the *representation matrices*, and we will refer to $d_\rho$ as the *degree* of the representation.

Representations play the role of basis functions, similar to that of sinusoids, in Fourier theory. The simplest basis functions are constant functions — and our first example of a representation is the *trivial* representation $\rho_0 : G \to \mathbb{R}$ which maps every element of $G$ to 1. As a more pertinent example, we define the *1st order permutation representation* of $S_n$ to be the degree $n$ representation, $\tau_1$, which maps a permutation $\sigma$ to its corresponding permutation matrix given by: $[\tau_1(\sigma)]_{ij} = \mathbb{1}\{\sigma(j) = i\}$. For example, the permutation in $S_3$ which swaps the second and third elements maps to:

$$\tau_1(1 \mapsto 1, 2 \mapsto 3, 3 \mapsto 2) = \begin{pmatrix} 1 & 0 & 0 \\ 0 & 0 & 1 \\ 0 & 1 & 0 \end{pmatrix}.$$

The $\tau_1$ representation can be thought of as a collection of $n^2$ functions at once, one for each matrix entry, $[\tau_1(\sigma)]_{ij}$. There are other possible permutation representations - for example the *2nd order unordered permutation representation*, $\tau_2$, is defined by the action of a permutation on unordered pairs of objects, $([\rho(\sigma)]_{\{i,j\},\{\ell,k\}} = \mathbb{1}\{\sigma(\{\ell, k\}) = \{i, j\}\})$, and is a degree $\frac{n(n-1)}{2}$ representation. And the list goes on to include many more complicated representations.

It is useful to think of two representations as being the same if the representation matrices are equal up to some consistent change of basis. This idea is formalized by declaring two representations $\rho$ and $\tau$ to be *equivalent* if there exists an invertible matrix $C$ such that $C^{-1} \cdot \rho(\sigma) \cdot C = \tau(\sigma)$ for all $\sigma \in G$. We write this as $\rho \equiv \tau$.

Most representations can be seen as having been built up by smaller representations. We say that a representation $\rho$ is *reducible* if there exist smaller representations $\rho_1, \rho_2$ such that $\rho \equiv \rho_1 \oplus \rho_2$ where $\oplus$ is defined to be the *direct sum representation*:

$$\rho_1 \oplus \rho_2(g) \triangleq \left( \begin{array}{c|c} \rho_1(g) & 0 \\ \hline 0 & \rho_2(g) \end{array} \right). \tag{1}$$

In general, there are infinitely many inequivalent representations. However, for any finite group, there is always a finite collection of atomic representations which can be used to build up any other representation using direct sums. These representations are referred to as the *irreducibles* of a group, and they are simply the collection of representations which are not reducible. We will refer to the set of irreducibles by $\mathcal{R}$. It can be shown that any representation of a finite group $G$ is equivalent to a direct sum of irreducibles [3], and hence, for any representation $\tau$, there exists a matrices $C$ for which $C^{-1} \cdot \tau \cdot C = \oplus_{\rho_i \in \mathcal{R}} \oplus \rho_i$, where the inner $\oplus$ refers to some finite number of copies of the irreducible $\rho_i$.

Describing the irreducibles of $S_n$ up to equivalence is a subject unto itself; We will simply say that there is a natural way to order the irreducibles of $S_n$ that corresponds to 'simplicity' in the same way that low frequency sinusoids are simpler than higher frequency ones. We will refer to the irreducibles in this order as $\rho_0, \rho_1, \ldots$. For example, the first two irreducibles form the first order permutation representation ($\tau_1 \equiv \rho_0 \oplus \rho_1$), and the second order permutation representation can be formed by the first 3 irreducibles.

Irreducible representation matrices are not always orthogonal, but they can always be chosen to be so (up to equivalence). For notational convenience, the irreducible representations in this paper will always be assumed to be orthogonal.

## 3.1 The Fourier transform

On the real line, the Fourier Transform corresponds to computing inner products of a function with sines and cosines at varying frequencies. The analogous definition for finite groups replaces the sinusoids by group representations.

**Definition 2.** Let $f : G \to \mathbb{R}$ be any function on a group $G$ and let $\rho$ be any representation on $G$. The *Fourier Transform* of $f$ at the representation $\rho$ is defined to be: $\hat{f}_\rho = \sum_\sigma f(\sigma)\rho(\sigma)$.

There are two important points which distinguish this Fourier Transform from the familiar version on the real line — it is matrix-valued, and instead of real numbers, the inputs to $\hat{f}$ are *representations* of $G$. The collection of Fourier Transforms of $f$ at all irreducibles form the Fourier Transform of $f$. As in the familiar case, there is an inverse transform given by:

$$f(\sigma) = \frac{1}{|G|} \sum_k d_{\rho_k} \mathrm{Tr} \left[ \hat{f}_{\rho_k}^T \cdot \rho_k(\sigma) \right], \tag{2}$$

where $k$ indexes over the collection of irreducibles of $G$.

We provide two examples for intuition. For functions on the real line, the Fourier Transform at zero gives the DC component of a signal. This is also true for functions on a group; If $f : G \to \mathbb{R}$ is any function, then the Fourier Transform of $f$ at the trivial representation is constant with $\hat{f}_{\rho_0} = \sum_\sigma f(\sigma)$. Thus, for any probability distribution $P$, we have $\hat{P}_{\rho_0} = 1$. If $P$ were the uniform distribution, then $\hat{P}_\rho = 0$ at all irreducibles except at the trivial representation.

The Fourier Transform at $\tau_1$ also has a simple interpretation:

$$[\hat{f}_{\tau_1}]_{ij} = \sum_{\sigma \in S_n} f(\sigma)[\tau_1(\sigma)]_{ij} = \sum_{\sigma \in S_n} f(\sigma)\mathbb{1}\{\sigma(j) = i\} = \sum_{\sigma:\sigma(j)=i} f(\sigma).$$

Thus, if $P$ is a distribution, then $\hat{P}_{\tau_1}$ is a matrix of marginal probabilties, where the $ij$-th element is the marginal probability that a random permutation drawn from $P$ maps element $j$ to $i$. Similarly, the Fourier transform of $P$ at the second order permutation representation is a matrix of marginal probabilities of the form $P(\sigma(\{i, j\}) = \{k, \ell\})$.

In Section 5, we will discuss function approximation by bandlimiting the Fourier coefficients, but this example should illustrate the fact that maintaining Fourier coefficients at low-order irreducibles is the same as maintaining low-order marginal probabilities, while higher order irreducibles correspond to more complicated marginals.

## 4 Inference in the Fourier domain

Bandlimiting allows for compactly storing a distribution over permutations, but the idea is rather moot if it becomes necessary to transform back to the primal domain each time an inference operation is called. Naively, the Fourier Transform on $S_n$ scales as $O((n!)^2)$, and even the fastest Fast Fourier Transforms for functions on $S_n$ are no faster than $O(n! \log(n!))$ (see [7] for example). To resolve this issue, we present a formulation of inference which operates solely in the Fourier domain, allowing us to avoid a costly transform. We begin by discussing exact inference in the Fourier domain, which is no more tractable than the original problem because there are $n!$ Fourier coefficients, but it will allow us to discuss the bandlimiting approximation in the next section. There are two operations to consider: *prediction/rollup*, and *conditioning*. The assumption for the rest of this section is that the Fourier Transforms of the transition and observation models are known. We discuss methods for obtaining the models in Section 7.

### 4.1 Fourier prediction/rollup

We will consider one particular type of transition model — that of a random walk over a group. This model assumes that $\sigma^{(t+1)}$ is generated from $\sigma^{(t)}$ by drawing a random permutation $\tau^{(t)}$ from some distribution $Q^{(t)}$ and setting $\sigma^{(t+1)} = \tau^{(t)} \sigma^{(t)}$. In our identity management example, $\tau^{(t)}$ represents a random identity permutation that might occur among tracks when they get close to each other (a *mixing event*), but the random walk model appears in other applications such as modeling card shuffles [3]. The Fourier domain Prediction/Rollup step is easily formulated using the convolution theorem (see also [3]):

**Proposition 3.** *Let $Q$ and $P$ be probability distributions on $S_n$. Define the convolution of $Q$ and $P$ to be the function $[Q * P](\sigma_1) = \sum_{\sigma_2} Q(\sigma_1 \cdot \sigma_2^{-1}) P(\sigma_2)$. Then for any representation $\rho$, $\left[\widehat{Q * P}\right]_\rho = \widehat{Q}_\rho \cdot \widehat{P}_\rho$, where the operation on the right side is matrix multiplication.*

The Prediction/Rollup step for the random walk transition model can be written as a convolution:

$$P(\sigma^{(t+1)}) = \sum_{\{(\sigma^{(t)}, \tau^{(t)}) \, : \, \sigma^{(t+1)} = \tau^{(t)} \cdot \sigma^{(t)}\}} Q^{(t)}(\tau^{(t)}) \cdot P(\sigma^{(t)}) = \sum_{\sigma^{(t)}} Q^{(t)}(\sigma^{(t+1)} \cdot (\sigma^{(t)})^{-1}) P(\sigma^{(t)}) = \left[Q^{(t)} * P\right](\sigma^{(t+1)}).$$

Then assuming that $\widehat{P}_\rho^{(t)}$ and $\widehat{Q}_\rho^{(t)}$ are given, the prediction/rollup update rule is simply:

$$\widehat{P}_\rho^{(t+1)} \leftarrow \widehat{Q}_\rho^{(t)} \cdot \widehat{P}_\rho^{(t)}.$$

Note that the update requires only knowledge of $\hat{P}$ and does not require $P$. Furthermore, the update is *pointwise* in the Fourier domain in the sense that the coefficients at the representation $\rho$ affect $\widehat{P}_\rho^{(t+1)}$ *only* at $\rho$.

### 4.2 Fourier conditioning

An application of Bayes rule to find a posterior distribution $P(\sigma|z)$ after observing some evidence $z$ requires a *pointwise product* of likelihood $L(z|\sigma)$ and prior $P(\sigma)$, followed by a normalization step. We showed earlier that the normalization constant $\sum_\sigma L(z|\sigma) \cdot P(\sigma)$ is given by the Fourier transform of $\widehat{L^{(t)} P^{(t)}}$ at the trivial representation — and therefore the normalization step of conditioning can be implemented by simply dividing each Fourier coefficient by the scalar $\left[\widehat{L^{(t)} P^{(t)}}\right]_{\rho_0}$.

The pointwise product of two functions $f$ and $g$, however, is trickier to formulate in the Fourier domain. For functions on the real line, the pointwise product of functions can be implemented by convolving the Fourier coefficients of $\hat{f}$ and $\hat{g}$, and so a natural question is: can we apply a similar operation for functions over other groups? Our answer to this is that there is an analogous (but more complicated) notion of convolution in the Fourier domain of a general finite group. We present a convolution-based conditioning algorithm which we call *Kronecker Conditioning*, which, in contrast to the pointwise nature of the Fourier Domain prediction/rollup step, and much like convolution, smears the information at an irreducible $\rho_k$ to other irreducibles.

**Fourier transforming the pointwise product**    Our approach to Fourier Transforming the point-wise product in terms of $\hat{f}$ and $\hat{g}$ is to manipulate the function $f(\sigma)g(\sigma)$ so that it can be seen as the result of an inverse Fourier Transform. Hence, the goal will be to find matrices $A_k$ (as a function of $\hat{f}, \hat{g}$) such that for any $\sigma \in G$,

$$f(\sigma) \cdot g(\sigma) = \frac{1}{|G|} \sum_k d_{\rho_k} \text{Tr} \left( A_k^T \cdot \rho_k(\sigma) \right), \tag{3}$$

where $A_k = \left[ \widehat{fg} \right]_{\rho_k}$. For any $\sigma \in G$ we can write the pointwise product in terms $\hat{f}$ and $\hat{g}$ using the inverse Fourier Transform (Equation 2):

$$
\begin{aligned}
f(\sigma) \cdot g(\sigma) &= \left[ \frac{1}{|G|} \sum_i d_{\rho_i} \text{Tr} \left( \hat{f}_{\rho_i}^T \cdot \rho_i(\sigma) \right) \right] \cdot \left[ \frac{1}{|G|} \sum_j d_{\rho_j} \text{Tr} \left( \hat{g}_{\rho_j}^T \cdot \rho_j(\sigma) \right) \right] \\
&= \left( \frac{1}{|G|} \right)^2 \sum_{i,j} d_{\rho_i} d_{\rho_j} \left[ \text{Tr} \left( \hat{f}_{\rho_i}^T \cdot \rho_i(\sigma) \right) \cdot \text{Tr} \left( \hat{g}_{\rho_j}^T \cdot \rho_j(\sigma) \right) \right].
\end{aligned}
\tag{4}
$$

Now we want to manipulate this product of traces in the last line to be just one trace (as in Equation 3), by appealing to some properties of the *matrix Kronecker product*. The connection to the pointwise product (first observed in [8]), lies in the property that for any matrices $U, V$, $\text{Tr}(U \otimes V) = (\text{Tr}\, U) \cdot (\text{Tr}\, V)$. Applying this to Equation 4, we have:

$$
\begin{aligned}
\text{Tr} \left( \hat{f}_{\rho_i}^T \cdot \rho_i(\sigma) \right) \cdot \text{Tr} \left( \hat{g}_{\rho_j}^T \cdot \rho_j(\sigma) \right) &= \text{Tr} \left( \left( \hat{f}_{\rho_i}^T \cdot \rho_i(\sigma) \right) \otimes \left( \hat{g}_{\rho_j}^T \cdot \rho_j(\sigma) \right) \right) \\
&= \text{Tr} \left( \left( \hat{f}_{\rho_i} \otimes \hat{g}_{\rho_j} \right)^T \cdot (\rho_i(\sigma) \otimes \rho_j(\sigma)) \right),
\end{aligned}
\tag{5}
$$

where the last line follows by standard matrix properties. The term on the right, $\rho_i(\sigma) \otimes \rho_j(\sigma)$, itself happens to be a representation, called the *Kronecker Product Representation*. In general, the Kronecker Product representation is reducible, and so it can decomposed into a direct sum of irreducibles. This means that if $\rho_i$ and $\rho_j$ are any two irreducibles of $G$, there exists a similarity transform $C_{ij}$ such that for any $\sigma \in G$,

$$C_{ij}^{-1} \cdot [\rho_i \otimes \rho_j](\sigma) \cdot C_{ij} = \bigoplus_k \bigoplus_{\ell=1}^{z_{ijk}} \rho_k(\sigma).$$

The $\oplus$ symbols here refer to a matrix direct sum as in Equation 1, $k$ indexes over all irreducible representations of $S_n$, while $\ell$ indexes over a number of *copies* of $\rho_k$ which appear in the decomposition. We index blocks on the right side of this equation by pairs of indices $(k, \ell)$. The number of copies of each $\rho_k$ is denoted by the integer $z_{ijk}$, the collection of which, taken over all triples $(i, j, k)$, are commonly referred to as the *Clebsch-Gordan* series. Note that we allow the $z_{ijk}$ to be zero, in which case $\rho_k$ does not contribute to the direct sum. The matrices $C_{ij}$ are known as the *Clebsch-Gordan coefficients*. The *Kronecker Product Decomposition* problem is that of finding the irreducible components of the Kronecker product representation, and thus to find the Clebsch-Gordan series/coefficients for each pair of representations $(\rho_i, \rho_j)$. Decomposing the Kronecker product inside Equation 5 using the Clebsch-Gordan series/coefficients yields the desired Fourier Transform, which we summarize here:

**Proposition 4.** *Let $\hat{f}, \hat{g}$ be the Fourier Transforms of functions $f$ and $g$ respectively, and for each ordered pair of irreducibles $(\rho_i, \rho_j)$, define the matrix: $A_{ij} \triangleq C_{ij}^{-1} \cdot \left( \hat{f}_{\rho_i} \otimes \hat{g}_{\rho_j} \right) \cdot C_{ij}$. Then the Fourier tranform of the pointwise product $fg$ is:*

$$\left[ \widehat{fg} \right]_{\rho_k} = \frac{1}{d_{\rho_k} |G|} \sum_{ij} d_{\rho_i} d_{\rho_j} \sum_{\ell=1}^{z_{ijk}} A_{ij}^{k\ell}, \tag{6}$$

*where $A_{ij}^{k\ell}$ is the block of $A_{ij}$ corresponding to the $(k, \ell)$ block in $\oplus_k \oplus_{\ell}^{z_{ijk}} \rho_k$.*

See the Appendix for a full proof of Proposition 4. The Clebsch-Gordan series, $z_{ijk}$, plays an important role in Equation 6, which says that the $(\rho_i, \rho_j)$ crossterm contributes to the pointwise product at $\rho_k$ *only* when $z_{ijk} > 0$. For example,

$$\rho_1 \otimes \rho_1 \equiv \rho_0 \oplus \rho_1 \oplus \rho_2 \oplus \rho_3. \tag{7}$$

So $z_{1,1,k} = 1$ for $k \leq 3$ and is zero otherwise.

Unfortunately, there are no analytical formulas for finding the Clebsch-Gordan series or coefficients, and in practice, these computations can take a long time. We emphasize however, that as fundamental quantities, like the digits of $\pi$, they need only be computed *once* and stored in a table for future reference. Due to space limitations, we will not provide complete details on computing these numbers. We refer the reader to Murnaghan [9], who provides general formulas for computing Clebsch-Gordan series for pairs of low-order irreducibles, and to Appendix 1 for details about computing Clebsch-Gordan coefficients. We will also make precomputed coefficients available on the web.

## 5 Approximate inference by bandlimiting

We approximate the probability distribution $P(\sigma)$ by fixing a bandlimit $B$ and maintaining the Fourier transform of $P$ only at irreducibles $\rho_0, \ldots \rho_B$. We refer to this set of irreducibles as $\mathcal{B}$. As on the real line, smooth functions are generally well approximated by only a few Fourier coefficients, while "wigglier" functions require more. For example, when $B = 3$, $\mathcal{B}$ is the set $\rho_0, \rho_1, \rho_2$, and $\rho_3$, which corresponds to maintaining marginal probabilities of the form $P(\sigma((i,j)) = (k,\ell))$. During inference, we follow the procedure outlined in the previous section but ignore the higher order terms which are not maintained. Pseudocode for bandlimited prediction/rollup and Kronecker conditioning is given in Figures 1 and 2.

Since the Prediction/Rollup step is pointwise in the Fourier domain, the update is exact for the maintained irreducibles because higher order irreducibles cannot affect those below the bandlimit. As in [5], we find that the error from bandlimiting creeps in through the conditioning step. For example, Equation 7 shows that if $B = 1$ (so that we maintain first-order marginals), then the pointwise product spreads information to second-order marginals. Conversely, pairs of higher-order irreducibles may propagate information to lower-order irreducibles. If a distribution is diffuse, then most of the energy is stored in low-order Fourier coefficients anyway, and so this is not a big problem. However, it is when the distribution is sharply concentrated at a small subset of permutations, that the low-order Fourier projection is unable to faithfully approximate the distribution, in many circumstances, resulting in a bandlimited Fourier Transform with negative "marginal probabilities"! To combat this problem, we present a method for enforcing nonnnegativity.

**Projecting to a relaxed marginal polytope**   The *marginal polytope*, $\mathcal{M}$, is the set of marginals which are consistent with some joint distribution over permutations. We project our approximation onto a relaxation of the marginal polytope, $\mathcal{M}'$, defined by linear inequality constraints that marginals be nonnegative, and linear equality constraints that they correspond to some legal Fourier transform. Intuitively, our relaxation produces matrices of marginals which are *doubly stochastic* (rows and columns sum to one and all entries are nonnegative), and satisfy lower-order marginal consistency (different high-order marginals are consistent at lower orders).

After each conditioning step, we apply a 'correction' to the approximate posterior $P^{(t)}$ by finding the bandlimited function in $\mathcal{M}'$ which is closest to $P^{(t)}$ in an $L_2$ sense. To perform the projection, we employ the Plancherel Theorem [3] which relates the $L_2$ distance between functions on $S_n$ to a distance metric in the Fourier domain.

**Proposition 5.**
$$\sum_{\sigma}(f(\sigma) - g(\sigma))^2 \quad = \quad \frac{1}{|G|}\sum_k d_{\rho_k} Tr\left(\left(\hat{f}_{\rho_k} - \hat{g}_{\rho_k}\right)^T \cdot \left(\hat{f}_{\rho_k} - \hat{g}_{\rho_k}\right)\right). \quad (8)$$

We formulate the optimization as a quadratic program where the objective is to minimize the right side of Equation 8 — the sum is taken only over the set of maintained irreducibles, $\mathcal{B}$, and subject to the linear constraints which define $\mathcal{M}'$.

We remark that even though the projection will always produce a Fourier transform corresponding to nonnegative marginals, there might not necessarily exist a joint probability distribution on $S_n$ consistent with those marginals. In the case of first-order marginals, however, the existence of a consistent joint distribution *is* guaranteed by the *Birkhoff-von Neumann* theorem [10], which states that a matrix is doubly stochastic *if and only if* it can be written as a convex combination of permutation matrices. And so for the case of first-order marginals, our relaxation is in fact, exact.

## 6 Related Work

The Identity Management problem was first introduced in [2] which maintains a doubly stochastic first order *belief matrix* to reason over data associations. Schumitsch et al. [4] exploits a similar idea, but formulated the problem in log-space.

Figure 1: Pseudocode for the Fourier Prediction/Rollup Algorithm.

PREDICTIONROLLUP
**foreach** $\rho_k \in \mathcal{B}$ **do** $\hat{P}_{\rho_k}^{(t+1)} \leftarrow \hat{Q}_{\rho_k}^{(t)} \cdot \hat{P}_{\rho_k}^{(t)}$ ;

Figure 2: Pseudocode for the Kronecker Conditioning Algorithm.

KRONECKERCONDITIONING
**foreach** $\rho_k \in \mathcal{B}$ **do** $\left[ \widehat{L^{(t)} P^{(t)}} \right]_{\rho_k} \leftarrow \mathbf{0}$  //Initialize Posterior
//Pointwise Product
**foreach** $\rho_i \in \mathcal{B}$ **do**
    **foreach** $\rho_j \in \mathcal{B}$ **do**
        $z \leftarrow CGseries(\rho_i, \rho_j)$ ;
        $C_{ij} \leftarrow CGcoefficients(\rho_i, \rho_j)$ ; $A_{ij} \leftarrow C_{ij}^T \cdot \left( \hat{f}_{\rho_i} \otimes \hat{g}_{\rho_j} \right) \cdot C_{ij}$ ;
        **for** $\rho_k \in \mathcal{B}$ such that $z_{ijk} \neq 0$ **do**
            **for** $\ell = 1$ **to** $z_k$ **do**
                $\left[ \widehat{L^{(t)} P^{(t)}} \right]_{\rho_k} \leftarrow \left[ \widehat{L^{(t)} P^{(t)}} \right]_{\rho_k} + \frac{d_{\rho_i} d_{\rho_j}}{d_{\rho_k} n!} A_{ij}^{k\ell}$  //$A_{ij}^{k\ell}$ is the $(k, \ell)$ block of $A_{ij}$
$Z \leftarrow \left[ \widehat{L^{(t)} P^{(t)}} \right]_{\rho_0}$ ;
**foreach** $\rho_k \in \mathcal{B}$ **do** $\left[ \widehat{L^{(t)} P^{(t)}} \right]_{\rho_k} \leftarrow \frac{1}{Z} \left[ \widehat{L^{(t)} P^{(t)}} \right]_{\rho_k}$  //Normalization

Kondor et al. [5] were the first to show that the data association problem could be approximately handled via the Fourier Transform. For conditioning, they exploit a modified FFT factorization which works on certain simplified observation models. Our approach generalizes the type of observations that can be handled in [5] and is equivalent in the simplified model that they present. We require $O(D^3 n^2)$ time in their setting. Their FFT method saves a factor of $D$ due to the fact that certain representation matrices can be shown to be sparse. Though we do not prove it, we observe that the Clebsch-Gordan coefficients, $C_{ij}$ are typically similarly sparse, which yields an equivalent running time in practice. In addition, Kondor et al. do not address the issue of projecting onto valid marginals, which, as we show in our experimental results, is fundamental in practice.

Willsky [8] was the first to formulate a nonabelian version of the FFT algorithm (for Metacyclic groups) as well as to note the connection between pointwise products and Kronecker product decompositions for general finite groups. In this paper, we address approximate inference, which is necessary given the $n!$ complexity of inference for the Symmetric group.

## 7  Experimental results

For small $n$, we compared our algorithm to exact inference on synthetic datasets in which tracks are drawn at random to be observed or swapped. For validation we measure the $L_1$ distance between true and approximate marginal distributions. In (Fig. 3(a)), we call several mixings followed by a single observation, after which we measured error. As expected, the Fourier approximation is better when there are either more mixing events, or when more Fourier coefficients are maintained. In (Fig. 3(b)) we allow for consecutive conditioning steps and we see that that the projection step is fundamental, especially when mixing events are rare, reducing the error dramatically. Comparing running times, it is clear that our algorithm scales gracefully compared to the exact solution (Fig. 3(c)).

We also evaluated our algorithm on data taken from a real network of 8 cameras (Fig. 3(d)). In the data, there are $n = 11$ people walking around a room in fairly close proximity. To handle the fact that people can freely leave and enter the room, we maintain a list of the tracks which are external to the room. Each time a new track leaves the room, it is added to the list and a mixing event is called to allow for $m^2$ pairwise swaps amongst the $m$ external tracks.

The number of mixing events is approximately the same as the number of observations. For each observation, the network returns a color histogram of the blob associated with one track. The task after conditioning on each observation is to predict identities for all tracks inside the room, and the evaluation metric is the fraction of accurate predictions. We compared against a baseline approach of predicting the identity of a track based on the most recently observed histogram at that track. This approach is expected to be accurate when there are many observations and discriminative appearance models, neither of which our problem afforded. As (Fig. 3(e)) shows,

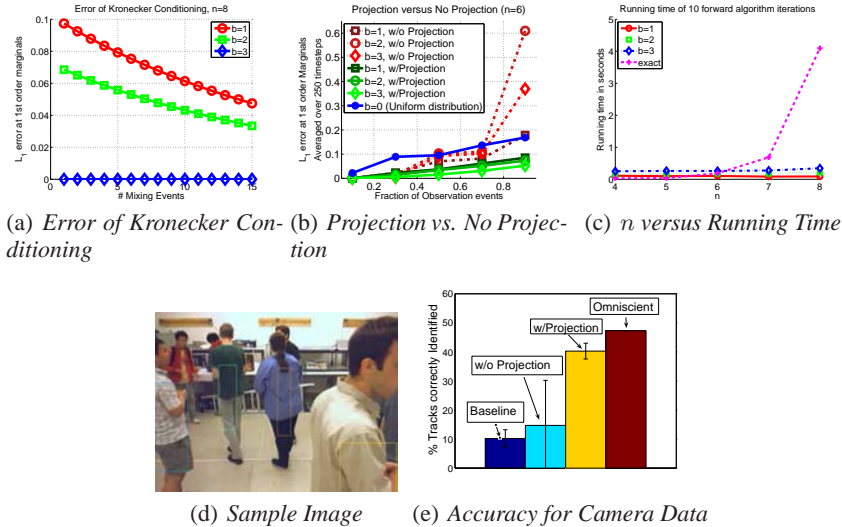

(a) *Error of Kronecker Conditioning*  (b) *Projection vs. No Projection*  (c) $n$ *versus Running Time*

(d) *Sample Image*  (e) *Accuracy for Camera Data*

Figure 3: Evaluation on synthetic ((a)-(c)) and real camera network ((d),(e)) data.

both the baseline and first order model(without projection) fared poorly, while the projection step dramatically boosted the accuracy. To illustrate the difficulty of predicting based on appearance alone, the rightmost bar reflects the performance of an *omniscient* tracker who knows the result of each mixing event and is therefore left only with the task of distinguishing between appearances.

## 8  Conclusions

We presented a formulation of hidden Markov model inference in the Fourier domain. In particular, we developed the Kronecker Conditioning algorithm which performs a convolution-like operation on Fourier coefficients to find the Fourier transform of the posterior distribution. We argued that bandlimited conditioning can result in Fourier coefficients which correspond to no distribution, but that the problem can be remedied by projecting to a relaxation of the marginal polytope. Our evaluation on data from a camera network shows that our methods outperform well when compared to the optimal solution in small problems, or to an omniscient tracker in large problems. Furthermore, we demonstrated that our projection step is fundamental to obtaining these high-quality results.

We conclude by remarking that the mathematical framework developed in this paper is quite general. In fact, both the prediction/rollup and conditioning formulations hold over any finite group, providing a principled method for approximate inference for problems with underlying group structure.

**Acknowledgments**
This work is supported in part by the ONR under MURI N000140710747, the ARO under grant W911NF-06-1-0275, the NSF under grants DGE-0333420, EEEC-540865, Nets-NOSS 0626151 and TF 0634803, and by the Pennsylvania Infrastructure Technology Alliance (PITA). Carlos Guestrin was also supported in part by an Alfred P. Sloan Fellowship. We thank Kyle Heath for helping with the camera data and Emre Oto, and Robert Hough for valuable discussions.

## References

[1] Y. Ivanov, A. Sorokin, C. Wren, and I. Kaur. Tracking people in mixed modality systems. Technical Report TR2007-11, MERL, 2007.

[2] J. Shin, L. Guibas, and F. Zhao. A distributed algorithm for managing multi-target identities in wireless ad-hoc sensor networks. In *IPSN*, 2003.

[3] P. Diaconis. *Group Representations in Probability and Statistics*. IMS Lecture Notes, 1988.

[4] B. Schumitsch, S. Thrun, G. Bradski, and K. Olukotun. The information-form data association filter. In *NIPS*. 2006.

[5] R. Kondor, A. Howard, and T. Jebara. Multi-object tracking with representations of the symmetric group. In *AISTATS*, 2007.

[6] X. Boyen and D. Koller. Tractable inference for complex stochastic processes. In *UAI*, 1998.

[7] R. Kondor. $\mathbb{S}_n$ob: a C++ library for fast Fourier transforms on the symmetric group, 2006. Available at http://www.cs.columbia.edu/~risi/Snob/.

[8] A. Willsky. On the algebraic structure of certain partially observable finite-state markov processes. *Information and Control*, 38:179–212, 1978.

[9] F.D. Murnaghan. The analysis of the kronecker product of irreducible representations of the symmetric group. *American Journal of Mathematics*, 60(3):761–784, 1938.

[10] J. van Lint and R.M. Wilson. *A Course in Combinatorics*. Cambridge University Press, 2001.

